# Very Fast EM-based Mixture Model Clustering using Multiresolution kd-trees

**Andrew W. Moore**

Robotics Institute, Carnegie Mellon University

Pittsburgh, PA 15213. awm@cs.cmu.edu

## Abstract

Clustering is important in many fields including manufacturing, biology, finance, and astronomy. Mixture models are a popular approach due to their statistical foundations, and EM is a very popular method for finding mixture models. EM, however, requires many accesses of the data, and thus has been dismissed as impractical (e.g. [9]) for data mining of enormous datasets. We present a new algorithm, based on the multiresolution kd-trees of [5], which dramatically reduces the cost of EM-based clustering, with savings rising linearly with the number of datapoints. Although presented here for maximum likelihood estimation of Gaussian mixture models, it is also applicable to non-Gaussian models (provided class densities are monotonic in Mahalanobis distance), mixed categorical/numeric clusters, and Bayesian methods such as Autoclass [1].

## 1   Learning Mixture Models

In a Gaussian mixture model (e.g. [3]), we assume that datapoints $\{\mathbf{x}_1 \ldots \mathbf{x}_R\}$ have been generated independently by the following process. For each $\mathbf{x}_i$ in turn, nature begins by randomly picking a class, $c_j$, from a discrete set of classes $\{c_1 \ldots c_N\}$. Then nature draws $\mathbf{x}_i$ from an $M$-dimensional Gaussian whose mean $\mu_j$ and covariance $\Sigma_j$ depend on the class. Thus we have

$$P(\mathbf{x}_i \mid c_j, \boldsymbol{\theta}) \sim ((2\pi)^M \|\Sigma_j\|)^{-1/2} \exp(-\frac{1}{2}(\mathbf{x}_i - \mu_j)^T \Sigma_j^{-1}(\mathbf{x}_i - \mu_j)) \qquad (1)$$

where $\boldsymbol{\theta}$ denotes all the parameters of the mixture: the class probabilities $p_j$ (where $p_j = P(c_j \mid \boldsymbol{\theta})$), the class centers $\mu_j$ and the class covariances $\Sigma_j$.

The job of a mixture model learner is to find a good estimate of the model, and Expectation Maximization (EM), also known as "Fuzzy $k$-means", is a popular

algorithm for doing so. The $t$th iteration of EM begins with an estimate $\theta^t$ of the model, and ends with an improved estimate $\theta^{t+1}$. Write

$$\theta^t = (p_1, \ldots p_N, \mu_1, \ldots \mu_N, \Sigma_1, \ldots, \Sigma_N) \qquad (2)$$

EM iterates over each point-class combination, computing for each class $c_j$ and each datapoint $\mathbf{x}_i$, the extent to which $\mathbf{x}_i$ is "owned" by $c_j$. The ownership is simply $w_{ij} = P(c_j \mid \mathbf{x}_i, \theta^t)$. Throughout this paper we will use the following notation:

$$\begin{aligned} a_{ij} &= P(\mathbf{x}_i \mid c_j, \theta^t) \\ w_{ij} &= P(c_j \mid \mathbf{x}_i, \theta^t) = a_{ij}p_j / \textstyle\sum_{k=1}^{N} a_{ik}p_k \text{(by Bayes' Rule)} \end{aligned}$$

Then the new value of the centroid, $\mu_j$, of the $j$th class in the new model $\theta^{t+1}$ is simply the weighted mean of all the datapoints, using the values $\{w_{1j}, w_{2j}, \ldots w_{Rj}\}$ as the weights. A similar weighted procedure gives the new estimates of the class probabilities and the class covariances:

$$p_j \leftarrow \frac{\text{sw}_j}{R} \quad, \quad \mu_j \leftarrow \frac{1}{\text{sw}_j} \sum_{i=1}^{R} w_{ij}\mathbf{x}_i \quad, \quad \Sigma_j \leftarrow \frac{1}{\text{sw}_j} \sum_{i=1}^{R} w_{ij}(\mathbf{x}_i - \mu_j)(\mathbf{x}_i - \mu_j)^T \quad (3)$$

where $\text{sw}_j = \sum_{i=1}^{R} w_{ij}$. Thus each iteration of EM visits every datapoint-class pair, meaning $NR$ evaluations of a $M$-dimensional Gaussian, and so needing $O(M^2 NR)$ arithmetic operations per iteration. This paper aims to reduce that cost.

An *mrkd*-tree (Multiresolution KD-tree), introduced in [2] and developed further in [5], is a binary tree in which each node is associated with a subset of the datapoints. The root node owns all the datapoints. Each non-leaf-node has two children, defined by a splitting dimension ND.SPLITDIM and a splitting value ND.SPLITVAL. The two children divide their parent's datapoints between them, with the left child owning those datapoints that are strictly less than the splitting value in the splitting dimension, and the right child owning the remainder of the parent's datapoints:

$$\begin{aligned} \mathbf{x}_i \in \text{ND.LEFT} &\quad\Leftrightarrow\quad \mathbf{x}_i[\text{ND.SPLITDIM}] < \text{ND.SPLITVAL} \text{ and } \mathbf{x}_i \in \text{ND} \qquad (4) \\ \mathbf{x}_i \in \text{ND.RIGHT} &\quad\Leftrightarrow\quad \mathbf{x}_i[\text{ND.SPLITDIM}] \geq \text{ND.SPLITVAL} \text{ and } \mathbf{x}_i \in \text{ND} \qquad (5) \end{aligned}$$

The distinguishing feature of *mrkd*-trees is that their nodes contain the following:

- ND.NUMPOINTS: The number of points owned by ND (equivalently, the average density in ND).
- ND.CENTROID: The centroid of the points owned by ND (equivalently, the first moment of the density below ND).
- ND.COV: The covariance of the points owned by ND (equivalently, the second moment of the density below ND).
- ND.HYPERRECT: The bounding hyper-rectangle of the points below ND

We construct *mrkd*-trees top-down, identifying the bounding box of the current node, and splitting in the center of the widest dimension. A node is declared to be a leaf, and is left unsplit, if the widest dimension of its bounding box is $\leq$ some threshold, $MBW$. If $MBW$ is zero, then all leaf nodes denote singleton or coincident points, the tree has $O(R)$ nodes and so requires $O(M^2 R)$ memory, and (with some care) the construction cost is $O(M^2 R + MR \log R)$. In practice, we set $MBW$ to 1% of the range of the datapoint components. The tree size and construction thus cost

considerably less than these bounds because in dense regions, tiny leaf nodes were able to summarize dozens of datapoints. Note too that the cost of tree-building is amortized—the tree must be built once, yet EM performs many iterations.

To perform an iteration of EM with the *mrk*d-tree, we call the function MAKESTATS (described below) on the root of the tree. MAKESTATS(ND, $\theta^t$) outputs $3N$ values: $(\text{sw}_1, \text{sw}_2, \ldots \text{sw}_N, \text{swx}_1, \ldots \text{swx}_N, \text{swxx}_1, \ldots \text{swxx}_N)$ where

$$\text{sw}_j = \sum_{\mathbf{x}_i \in \text{ND}} w_{ij} \quad , \quad \text{swx}_j = \sum_{\mathbf{x}_i \in \text{ND}} w_{ij} \mathbf{x}_i \quad , \quad \text{swxx}_j = \sum_{\mathbf{x}_i \in \text{ND}} w_{ij} \mathbf{x}_i \mathbf{x}_i^T \quad (6)$$

The results of MAKESTATS(ROOT) provide sufficient statistics to construct $\theta^{t+1}$:

$$p_j \leftarrow \text{sw}_j / R \quad , \quad \mu_j \leftarrow \text{swx}_j / \text{sw}_j \quad , \quad \Sigma_j \leftarrow (\text{swxx}_j / \text{sw}_j) - \mu_j \mu_j^T \quad (7)$$

If MAKESTATS is called on a leaf node, we simply compute, for each $j$,

$$\bar{w}_j = P(c_j \mid \bar{\mathbf{x}}, \theta^t) = P(\bar{\mathbf{x}} \mid c_j, \theta^t) P(c_j \mid \theta^t) / \sum_{k=1}^{N} P(\bar{\mathbf{x}} \mid c_k, \theta^t) P(c_k \mid \theta^t) \quad (8)$$

where $\bar{\mathbf{x}}$ = ND.CENTROID, and where all the items in the right hand equation are easily computed. We then return $\text{sw}_j = \bar{w}_j \times$ ND.NUMPOINTS, $\text{swx}_j = \bar{w}_j \times$ ND.NUMPOINTS $\times \bar{\mathbf{x}}$ and $\text{swxx}_j = \bar{w}_j \times$ ND.NUMPOINTS $\times$ ND.COV. The reason we can do this is that, if the leaf node is very small, there will be little variation in $w_{ij}$ for the points owned by the node and so, for example $\sum w_{ij} \mathbf{x}_i \approx \bar{w}_j \sum \mathbf{x}_i$. In the experiments below we use very tiny leaf nodes, ensuring accuracy.

If MAKESTATS is called on a non-leaf-node, it can easily compute its answer by recursively calling MAKESTATS on its two children and then returning the sum of the two sets of answers. In general, that is exactly how we will proceed. If that was the end of the story, we would have little computational improvement over conventional EM, because one pass would fully traverse the tree, which contains $O(R)$ nodes, doing $O(NM^2)$ work per node.

We will win if we ever spot that, at some intermediate node, we can *prune*, i.e. evaluate the node as if it were a leaf, without searching its descendents, but without introducing significant error into the computation.

To do this, we will compute, for each $j$, the minimum and maximum $w_{ij}$ that any point inside the node could have. This procedure is more complex than in the case of locally weighted regression [5].

We wish to compute $w_j^{\text{min}}$ and $w_j^{\text{max}}$ for each $j$, where $w_j^{\text{min}}$ is a lower bound on $\min_{\mathbf{x}_i \in \text{ND}} w_{ij}$ and $w_j^{\text{max}}$ is an upper bound on $\max_{\mathbf{x}_i \in \text{ND}} w_{ij}$. This is hard because $w_j^{\text{min}}$ is determined not only by the mean and covariance of the $j$th class but also the other classes. For example, in Figure 1, $w_{32}$ is approximately 0.5, but it would be much larger if $c_1$ were further to the left, or had a thinner covariance.

But remember that the $w_{ij}$'s are defined in terms of $a_{ij}$'s, thus: $w_{ij} = a_{ij} p_j / \sum_{k=1}^{N} a_{ik} p_k$. We *can* put bounds on the $a_{ij}$'s relatively easily. It simply requires that for each $j$ we compute[1] the closest and furthest point from $\mu_j$ within

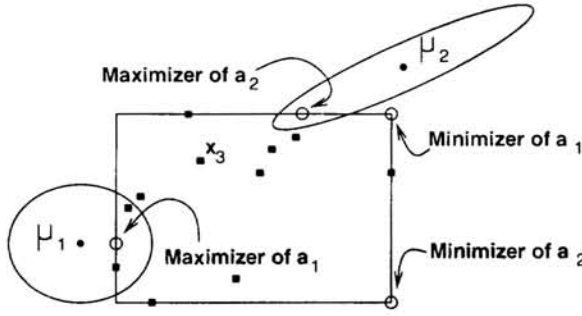

Figure 1: The rectangle denotes a hyper-rectangle in the mrkd-tree. The small squares denote datapoints "owned" by the node. Suppose there are just two classes, with the given means, and co-variances depicted by the ellipses. Small circles indicate the locations within the node for which $a_j$ (i.e. $P(x \mid c_j)$) would be extremized.

ND.HYPERRECT, using the Mahalanobis distance $MHD(\mathbf{x}, \mathbf{x}') = (\mathbf{x} - \mathbf{x}')^T \Sigma_j^{-1} (\mathbf{x} - \mathbf{x}')$. Call these shortest and furthest squared distances $MHD^{\min}$ and $MHD^{\max}$. Then

$$a_j^{\min} = ((2\pi)^M \|\Sigma_j\|)^{-1/2} \exp(-\frac{1}{2} MHD^{\max}) \qquad (9)$$

is a lower bound for $\min_{\mathbf{x}_i \in \text{ND}} a_{ij}$, with a similar definition of $a_j^{\max}$. Then write

$$\min_{\mathbf{x}_i \in \text{ND}} w_{ij} = \min_{\mathbf{x}_i \in \text{ND}} (a_{ij} p_j / \sum_k a_{ik} p_k) = \min_{\mathbf{x}_i \in \text{ND}} (a_{ij} p_j / (a_{ij} p_j + \sum_{k \neq j} a_{ik} p_k))$$

$$\geq a_j^{\min} p_j / (a_j^{\min} p_j + \sum_{k \neq j} a_k^{\max} p_k) = w_j^{\min}$$

where $w_j^{\min}$ is our lower bound. There is a similar definition for $w_j^{\max}$. The inequality is proved by elementary algebra, and requires that all quantities are positive (which they are). We can often tighten the bounds further using a procedure that exploits the fact that $\sum_j w_{ij} = 1$, but space does not permit further discussion.

We will prune if $w_j^{\min}$ and $w_j^{\max}$ are close for all $j$. What should be the criterion for closeness? The first idea that springs to mind is: Prune if $\forall j . (w_j^{\max} - w_j^{\min} < \epsilon)$. But such a simple criterion is not suitable: some classes may be accumulating very large sums of weights, whilst others may be accumulating very small sums. The large-sum-weight classes can tolerate far looser bounds than the small-sum-weight classes. Here, then, is a more satisfactory pruning criterion: Prune if $\forall j . (w_j^{\max} - w_j^{\min} < \tau w_j^{\text{total}})$ where $w_j^{\text{total}}$ is the total weight awarded to class $j$ over the entire dataset, and $\tau$ is some small constant. Sadly, $w_j^{\text{total}}$ is not known in advance, but happily we can find a lower bound on $w_j^{\text{total}}$ of $w_j^{\text{sofar}} + \text{ND.NUMPOINTS} \times w_j^{\min}$, where $w_j^{\text{sofar}}$ is the total weight awarded to class $j$ so far during the search over the kd-tree.

The algorithm as described so far performs divide-and-conquer-with-cutoffs on the set of datapoints. In addition, it is possible to achieve an extra acceleration by means of divide and conquer on the class centers. Suppose there were $N = 100$ classes. Instead of considering all 100 classes at all nodes, it is frequently possible to determine at some node that the maximum possible weight $w_j^{\max}$ for some class $j$ is less than a miniscule fraction of the minimum possible weight $w_k^{\min}$ for some other class $k$. Thus if we ever find that in some node $w_j^{\max} < \lambda w_k^{\min}$ where $\lambda = 10^{-4}$, then class $c_j$ is removed from consideration from all descendents of the current node. Frequently this means that near the tree's leaves, only a tiny fraction of the classes compete for ownership of the datapoints, and this leads to large time savings.

## 2 Results

We have subjected this approach to numerous Monte-Carlo empirical tests. Here we report on one set of such tests, created with the following methodology.

- We randomly generate a mixture of Gaussians in $M$-dimensional space (by default $M = 2$). The number of Gaussians, $N$ is, by default, 20. Each Gaussian has a mean lying within the unit hypercube, and a covariance matrix randomly generated with diagonal elements between 0 up to $4\sigma^2$ (by default, $\sigma = 0.05$) and random non-diagonal elements that ensure symmetric positive definiteness. Thus the distance from a Gaussian center to its 1-standard-deviation contour is of the order of magnitude of $\sigma$.
- We randomly generate a dataset from the mixture model. The number of points, $R$, is (by default) 160,000. Figure 2 shows a typical generated set of Gaussians and datapoints.
- We then build an *mrk*d-tree for the dataset, and record the memory requirements and real time to build (on a Pentium 200Mhz, in seconds).
- We then run EM on the data. EM begins with an entirely different set of Gaussians, randomly generated using the same procedure.
- We run 5 iterations of the conventional EM algorithm and the new *mrk*d-tree-based algorithm. The new algorithm uses a default value of 0.1 for $\tau$. We record the real time (in seconds) for each iteration of each algorithm, and we also record the mean log-likelihood score $(1/R) \sum_{i=1}^{R} \log P(\mathbf{x}_i \mid \boldsymbol{\theta}^t)$ for the $t$th model for both algorithms.

Figure 3 shows the nodes that are visited during Iteration 2 of the Fast EM with $N = 6$ classes. Table 1 shows the detailed results as the experimental parameters are varied. Speedups vary from 8-fold to 1000-fold. There are 100-fold speedups even with very wide (non-local) Gaussians. In other experiments, similar results were also obtained on real datasets that disobey the Gaussian assumption. There too, we find one- and two-order-of-magnitude computational advantages with indistinguishable statistical behavior (no better and no worse) compared with conventional EM.

**Real Data:** Preliminary experiments in applying this to large datasets have been encouraging. For three-dimensional galaxy clustering with 800,000 galaxies and 1000 clusters, traditional EM needed 35 minutes per iteration, while the *mrk*d-trees required only 13 seconds. With 1.6 million galaxies, traditional EM needed 70 minutes and *mrk*d-trees required 14 seconds.

## 3 Conclusion

The use of variable resolution structures for clustering has been suggested in many places (e.g. [7, 8, 4, 9]). The BIRCH system, in particular, is popular in the database community. BIRCH is, however, unable to identify second-moment features of clusters (such as non-axis-aligned spread). Our contributions have been the use of a multi-resolution approach, with associated computational benefits, and the introduction of an efficient algorithm that leaves the statistical aspects of mixture model estimation unchanged. The growth of recent data mining algorihms that are *not* based on statistical foundations has freqently been justified by the following statement: Using state-of-the-art statistical techniques is too expensive because such techniques were not designed to handle large datasets and become intractable with millions of datapoints. In earlier work we provided evidence that this statement may

| | |
|---|---|
| Effect of **Number of Datapoints**, $R$:<br>As $R$ increases so does the computational advantage, essentially linearly. The tree-build time (11 seconds at worst) is a tiny cost compared with even just one iteration of Regular EM (2385 seconds, on the big dataset.) **FinalSlowSecs:** 2385. **FinalFastSecs:** 3. | 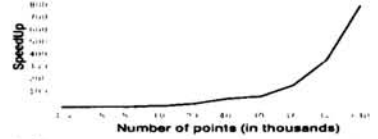 |
| Effect of **Number of Dimensions**, $M$:<br>As with many $k$d-tree algorithms, the benefits decline as dimensionality increases, yet even in 6 dimensions, there is an 8-fold advantage. **FinalSlowSecs:** 2742. **FinalFastSecs:** 310.25. | 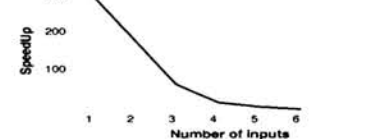 |
| Effect of **Number of Classes**, $N$:<br>Conventional EM slows down linearly with the number of classes. Fast EM is clearly sublinear, with a 70-fold speedup even with 320 classes. Note how the tree size grows. This is because more classes mean a more uniform data distribution and fewer datapoints "sharing" tree leaves. **FinalSlowSecs:** 9278. **FinalFastSecs:** 143.3. |  |
| Effect of **Tau**, $\tau$:<br>The larger $\tau$, the more willing we are to prune during the tree search, and thus the faster we search, but the less accurately we mirror EM's statistical behavior. Indeed when $\tau$ is large, the discrepancy in the log likelihood is relatively large. **FinalSlowSecs:** 584.5. **FinalFastSecs:** 2. | 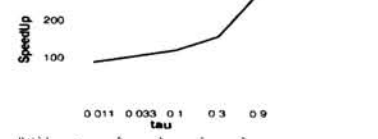 |
| Effect of **Standard Deviation**, $\sigma$:<br>Even with very wide Gaussians, with wide support, we still get large savings. The nodes that are pruned in these cases are rarely nodes with one class owning all the probability, but instead are nodes where all classes have non-zero, but little varying, probability. **FinalSlowSecs:** 583. **FinalFastSecs:** 4.75. | 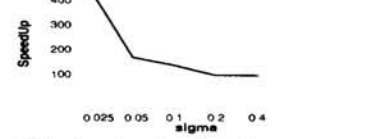 |

Table 1: In all the above results all parameters were held at their default values except for one, which varied as shown in the graphs. Each graph shows the factor by which the new EM is faster than the conventional EM. Below each graph is the time to build the $mr$kd-tree in seconds and the number of nodes in the tree. Note that although the tree building cost is not included in the speedup calculation, it is negligible in all cases, especially considering that only one tree build is needed for all EM iterations. Does the approximate nature of this process result in inferior clusters? The answer is no: the quality of clusters is indistinguishable between the slow and fast methods when measured by log-likelihood and when viewed visually.

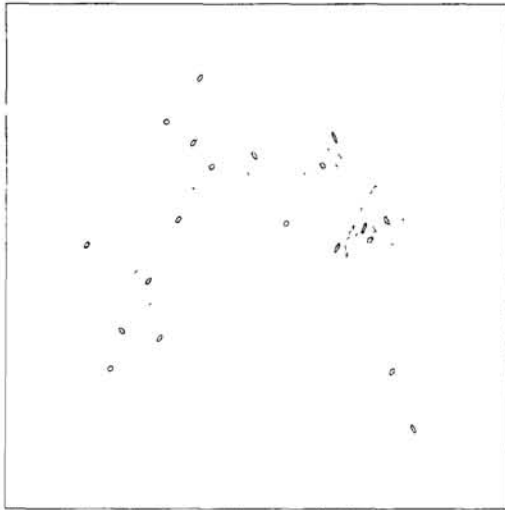

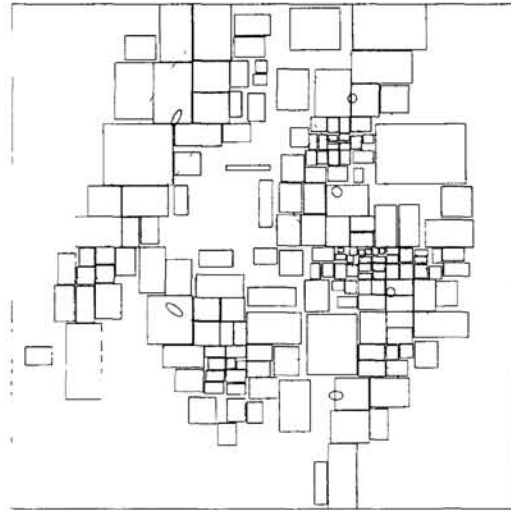

Figure 2: A typical set of Gaussians generated by our random procedure. They in turn generate the datasets upon which we compare the performance of the old and new implementations of EM.

Figure 3: The ellipses show the model $\theta^t$ at the *start* of an EM iteration. The rectangles depict the *mrkd*-tree nodes that were pruned. Observe larger rectangles (and larger savings) in areas with less variation in class probabilities. Note this is not merely able to only prune where the data density is low.

not apply for locally weighted regression [5] or Bayesian network learning [6], and we hope this paper provides some evidence that it also needn't apply to clustering.

## Footnotes

[1] Computing these points requires non-trivial computational geometry because the covariance matrices are not necessarily axis-aligned. There is no space here for details.

# References

[1] P. Cheeseman and R. Oldford. *Selecting Models from Data: Artificial Intelligence and Statistics IV. Lecture Notes in Statistics, vol. 89.* Springer Verlag, 1994.

[2] K. Deng and A. W. Moore. Multiresolution Instance-based Learning. In *Proceedings of IJCAI-95.* Morgan Kaufmann, 1995.

[3] R. O. Duda and P. E. Hart. *Pattern Classification and Scene Analysis.* John Wiley & Sons, 1973.

[4] M. Ester, H. P. Kriegel, and X. Xu. A Database Interface for Clustering in Large Spatial Databases. In *Proceedings of the First International Conference on Knowledge Discovery and Data Mining.* AAAI Press, 1995.

[5] A. W. Moore, J. Schneider, and K. Deng. Efficient Locally Weighted Polynomial Regression Predictions. In D. Fisher, editor, *Proceedings of the 1997 International Machine Learning Conference.* Morgan Kaufmann, 1997.

[6] Andrew W. Moore and M. S. Lee. Cached Sufficient Statistics for Efficient Machine Learning with Large Datasets. *Journal of Artificial Intelligence Research*, 8, March 1998.

[7] S. M. Omohundro. Efficient Algorithms with Neural Network Behaviour. *Journal of Complex Systems*, 1(2):273–347, 1987.

[8] S. M. Omohundro. Bumptrees for Efficient Function, Constraint, and Classification Learning. In R. P. Lippmann, J. E. Moody. and D. S. Touretzky, editors, *Advances in Neural Information Processing Systems 3.* Morgan Kaufmann, 1991.

[9] T. Zhang, R. Ramakrishnan, and M. Livny. BIRCH: An Efficient Data Clustering Method for Very Large Databases. In *Proceedings of the Fifteenth ACM SIGACT-SIGMOD-SIGART Symposium on Principles of Database Systems : PODS 1996.* Assn for Computing Machinery, 1996.
